# Fast Algorithms for Large-State-Space HMMs with Applications to Web Usage Analysis

**Pedro F. Felzenszwalb[1], Daniel P. Huttenlocher[2], Jon M. Kleinberg[2]**

[1]AI Lab, MIT, Cambridge MA 02139
[2]Computer Science Dept., Cornell University, Ithaca NY 14853

## Abstract

In applying Hidden Markov Models to the analysis of massive data streams, it is often necessary to use an artificially reduced set of states; this is due in large part to the fact that the basic HMM estimation algorithms have a quadratic dependence on the size of the state set. We present algorithms that reduce this computational bottleneck to linear or near-linear time, when the states can be embedded in an underlying grid of parameters. This type of state representation arises in many domains; in particular, we show an application to traffic analysis at a high-volume Web site.

## 1 Introduction

Hidden Markov Models (HMMs) are used in a wide variety of applications where a sequence of observable events is correlated with or caused by a sequence of unobservable underlying states (e.g., [8]). Despite their broad applicability, HMMs are in practice limited to problems where the number of hidden states is relatively small. The most natural such problems are those where some abstract categorization provides a small set of discrete states, such as phonemes in the case of speech recognition or coding and structure in the case of genomics. Recently, however, issues arising in massive data streams, such as the analysis of usage logs at high-traffic Web sites, have led to problems that call naturally for HMMs with large state sets over very long input sequences.

A major obstacle in scaling HMMs up to larger state spaces is the computational cost of implementing the basic primitives associated with them: given an $n$-state HMM and a sequence of $T$ observations, determining the probability of the observations, or the state sequence of maximum probability, takes $O(Tn^2)$ time using the forward-backward and Viterbi algorithms. The quadratic dependence on the number of states is a long-standing bottleneck that necessitates a small (often artificially coarsened) state set, particularly when the length $T$ of the input is large.

In this paper, we present algorithms that overcome this obstacle for a broad class of HMMs. We improve the running times of the basic estimation and inference primitives to have a linear or near-linear dependence on the number of states, for a family of models in which the states are embedded as discrete grid points in an underlying parameter space, and the state transition costs (the negative logs of

the state transition probabilities) correspond to a possibly non-metric distance on this space. This kind of *embedded-state model* arises in many domains, including object tracking, de-noising one-dimensional signals, and event detection in time series. Thus the algorithms can be seen as extending the applicability of HMMs to problems that are traditionally solved with more restricted linear Gaussian state-space models such as Kalman filtering. Non-Gaussian state-space techniques are a research focus in their own right (e.g., [6]) and our methods could be used to improve their efficiency.

Given a structured embedding of states in an underlying $d$-dimensional space, our approach is to reduce the amount of work in the dynamic programming iterations of the Viterbi and forward-backward algorithms. For the Viterbi algorithm, we make use of *distance transform* (also known as *Voronoi surface*) techniques, which are widely used in computer vision, image processing, and discrete computational geometry [2]. For a broad class of distance functions on the embedding space (including functions that are far from obeying the triangle inequality), we are able to run each dynamic programming step of the Viterbi algorithm in $O(n)$ time, yielding an overall running time of $O(Tn)$. In the case of the forward-backward algorithm, we are able to achieve $O(Tn)$ time for any transition probabilities that can be decomposed into a constant number of *box filters* [10]. Box filters are discrete convolution kernels that can be computed in linear time; many functions, including the Gaussian, can be expressed or approximated as the composition of a few box filters. Moreover, in the case of the forward-backward algorithm, we are able to obtain a running time of $O(Tn \log n)$ for arbitrary state transition probabilities, as long as they are based only on differences in the embedded positions of the states.

A motivating application for our work comes from the analysis of Web usage data [1]. We focus on the Internet Archive site (www.archive.org) as a prototypical example of a high-traffic site (millions of page-visits per month) offering an array of digital items for download. An important question at such a site is to determine variations in user interest in the items being offered. We use a coin-tossing HMM model in which the discrete states correspond to the current probability of a user downloading a given item; this state set has a natural embedding in the interval $[0, 1]$. We study the effect of increasing the number of states, and find that a fairly large state set (of size roughly a hundred or more) is needed in order to detect brief but significant events that affect the download rate. With tens of millions of observations and a state set of this size, practical analysis would be computationally prohibitive without the faster HMM algorithms described here.

It should be noted that our methods can also be used in belief revision and belief propagation algorithms for Bayesian networks (e.g., [7]), as these algorithms are essentially variants of the Viterbi and forward-backward algorithms for HMMs. The methods are also applicable to continuous Markov models, which have recently been employed for Web user modeling based on duration of page views [9].

## 2  Hidden Markov Models

We briefly review HMMs; however we assume that the reader is familiar both with HMMs and with the Viterbi and forward-backward estimation algorithms. Rabiner [8] provides a good introduction to HMMs; we use notation similar to his. An HMM can be represented by a 5-tuple $\lambda = (S, V, A, B, \pi)$ where $S = \{s_1, \ldots, s_n\}$ is a finite set of (hidden) states, $V = \{v_1, \ldots, v_m\}$ is a finite set of observable symbols, $A$ is an $n \times n$ matrix with entries $a_{ij}$ corresponding to the probability of going from state $i$ to state $j$, $B = \{b_i(k)\}$ where $b_i(k)$ specifies the probability of observing symbol $v_k$ in state $s_i$, and $\pi$ is an $n$-vector with each entry $\pi_i$ corresponding to the probability

| Function | Viterbi | Forward-Backward |
|---|---|---|
| $a_{ij} = p$ if $|i-j| \le d$, $a_{ij} = 0$ otherwise | Min-filter | Box sum |
| $a_{ij} \propto \exp(-|i-j|^2/2\sigma^2)$ | $L_2^2$ dist. trans. | Gaussian approx. |
| $a_{ij} \propto \exp(-k|i-j|)$ | $L_1$ dist. trans. | FFT |
| $a_{ij} = p$ if $|i-j| \le d$, $a_{ij} = q$ otherwise | Combin. min-filter | Combin. box sum |
| $a_{ij} \propto \exp(-|i-j|^2/2\sigma^2)$ if $|i-j| \le d$, $a_{ij} \propto \exp(-k|i-j|)$ otherwise | Combin. dist. trans. | FFT |

Table 1: Some transition probabilities that can be handled efficiently using our techniques (see text for an explanation). All running times are $O(Tn)$ except those using the FFT which are $O(Tn \log n)$.

that the initial state of the system is $s_i$.

Let $q_t$ denote the state of the system at time $t$, while $o_t$ denotes the observed symbol at time $t$. Given a sequence of observations $O = (o_1, \ldots, o_T)$ there are three standard estimation (or inference) problems that have wide applications:

1. Find a state sequence $Q = (q_1, \ldots, q_T)$ maximizing $P(Q|O, \lambda)$.

2. Compute $P(O|\lambda)$, the probability of an observation sequence being generated by $\lambda$.

3. Compute the posterior probabilities of each state, $P(q_t = s_i|O, \lambda)$.

As is well known these problems can be solved in $O(Tn^2)$ time using the Viterbi algorithm for the first task and the forward-backward algorithm for the others. We show how to solve them more efficiently for a wide range of transition probabilities based on *differences* between states that are embedded in an underlying grid. This grid can be multi-dimensional, however in this paper we consider only the one-dimensional case. Table 1 lists some widely applicable transition probability distributions that can be handled by our methods. The algorithms for each distribution differ slightly and are explained in the subsequent sections. The distributions given in the bottom part of the table can be computed as combinations of the basic distributions in the top part. Other distributions can be obtained using these same combination techniques, as long as only a constant number of distributions are being combined.

An additional problem, which we do not explicitly consider here, is that of determining the best model $\lambda$ given some set of observed sequences $\{O_1, \ldots, O_l\}$. However the most widely used technique for solving this problem, expectation maximization (EM), requires repeatedly running the forward-backward algorithm. Thus our algorithms also indirectly make the model learning problem more efficient.

## 2.1 Viterbi Algorithm

The Viterbi algorithm is used to find a maximum posterior probability state sequence, that is a sequence $Q = (q_1, \ldots, q_T)$ maximizing $P(Q|O, \lambda)$. The main computation is to determine the highest probability along a path, accounting for the observations and ending in a given state. While there are an exponential number of possible paths, the Viterbi algorithm uses a dynamic programming approach

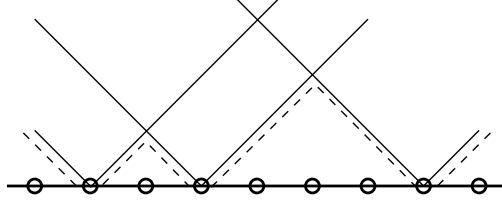

Figure 1: An example of the $L_1$ distance transform for a grid with $n = 9$ points containing the point set $P = \{1, 3, 7\}$. The distance transform value at each point is given by the height of the lower envelope, depicted as a dashed contour.

(see e.g., [8]), employing the recursive equation

$$\delta_{t+1}(j) = b_j(o_{t+1}) \max_i \left(\delta_t(i)a_{ij}\right),$$

where $\delta_t(i)$, for $i = 1, 2, \ldots, n$, encodes the highest probability along a path which accounts for the first $t$ observations and ends in state $s_i$. The maximization term takes $O(n^2)$ time, resulting in an overall time of $O(Tn^2)$ for a sequence of length $T$. Computing $\delta_t$ for each time step is only the first pass of the Viterbi algorithm. In a subsequent backward pass, a minimizing path is found. This takes only $O(Tn)$ time, so the forward computation is the dominant part of the running time.

In general a variant of the Viterbi algorithm is employed that uses negative log probabilities rather than probabilities, such that the computation becomes $\delta'_{t+1}(j) = b'_j(o_{t+1}) + \min_i(\delta'_t(i) + a'_{ij})$, where $'$ is used to denote a negative log probability. We now turn to the computation of $\delta'$ for restricted forms of the transition costs $a'_{ij}$, where there is an underlying parameter space such that the costs can be expressed in terms of a distance between parameter values corresponding to the states. Let us denote such cost functions by $\rho(i - j)$. Then,

$$\delta'_{t+1}(j) = b'_j(o_{t+1}) + \min_i \left(\delta'_t(i) + \rho(i - j)\right). \tag{1}$$

We now show how the minimization in the second term can be computed in $O(n)$ time rather than $O(n^2)$. The approach is based on a generalization of the *distance transform*, which is defined for sets of points on a grid. Consider a grid with $N$ locations and a point set $P$ on that grid. The distance transform of $P$ specifies for each grid location, the distance to the closest point in the set $P$,

$$D_P(j) = \min_{i \in P} \rho(i - j).$$

Clearly the distance transform can be computed in $O(N^2)$ time by considering all pairs of grid locations. However, it can also be computed in linear time for many distance functions using simple algorithms (e.g., [2, 5]). These algorithms have small constants and are fast in practice. The algorithms work for distance transforms of $d$-dimensional grids, not just for the one-dimensional case that we illustrate here.

In order to compute the distance transform efficiently it is commonly expressed as,

$$D_P(j) = \min_i \left(\rho(i - j) + 1(i)\right),$$

where $1(i)$ is an indicator function for the set $P$ such that $1(i) = 0$ when $i \in P$ and $1(i) = \infty$ otherwise. Intuitively one can think of a collection of upward facing cones, one rooted at each grid location that is in the set $P$. The transform is then obtained by taking the lower envelope (or minimum) of these cones. For concreteness consider

the one-dimensional case with the $L_1$ distance between grid locations. In this case the "cones" are v-shapes of slope 1 rising from the value $y = 0$ at each grid location that corresponds to a point of the set $P$, as illustrated in Figure 1.

It is straightforward to verify that a simple two-pass algorithm correctly computes this one-dimensional distance transform. First the vector $D(j)$ is initialized to $1(j)$. Then in the forward pass, each successive element of $D(j)$ is set to the minimum of its own value and one plus the value of the previous element (this is done "in place" so that updates affect one another).

$$j = 1, ..., n - 1 : \ D(j) = \min(D(j), D(j - 1) + 1).$$

The backward pass is analogous,

$$j = n - 2, ..., 0 : \ D(j) = \min(D(j), D(j + 1) + 1).$$

Consider the example in Figure 1. After the initialization step the value of $D$ is $(\infty, 0, \infty, 0, \infty, \infty, \infty, 0, \infty)$, after the forward pass it is $(\infty, 0, 1, 0, 1, 2, 3, 0, 1)$ and after the backward pass the final answer of $(1, 0, 1, 0, 1, 2, 1, 0, 1)$.

This computation of the distance transform does not depend on the form of the function $1(i)$. This suggests a generalization of distance transforms where the indicator function $1(i)$ is replaced with an arbitrary function,

$$D_f(j) = \min_i \left( \rho(i - j) + f(i) \right).$$

The same observation was used in [4] to efficiently compute certain tree-based cost functions for visual recognition of multi-part objects. Intuitively, the upward-facing cones are now rooted at height $f(i)$ rather than at zero, and are positioned at every grid location. The function $D_f$ is as above the lower envelope of these cones.

This generalized distance transform $D_f$ is precisely the form of the minimization term in the computation of the Viterbi recursion $\delta'$ in equation (1), where each state corresponds to a grid point. The algorithm above can be used to compute each step of the Viterbi minimization in $O(n)$ time when $\rho$ is the $L_1$ norm, giving an $O(Tn)$ algorithm overall. This corresponds to the problem in the third row of Table 1. The computation for the second row of the table is similar, except that computing the distance transform for the $L_2$ distance squared is a bit more involved (see [5]). The distribution in the first row of the table can be handled using a linear time algorithm for the min-filter [3].

Combinations of transforms can be formed by computing each function separately and then taking the minimum of the results. The entries in the bottom part of Table 1 show two such combinations. The function in the fourth row is often of practical interest, where the probability is $p$ of staying near the current state and $q$ of transitioning to any other state. The function in the last row is a so-called "truncated quadratic", arising commonly in robust statistics. In the experimental section we use a similar function that is the combination of two linear components with different slopes.

## 2.2   Forward-Backward Algorithm

The forward-backward algorithm is used to find the probability of the observed sequence given the the model, $P(O|\lambda)$. The computation also determines the posterior probability of the states at each time, $P(q_t = s_i|O, \lambda)$. Most of the work in the forward-backward algorithm is spent in determining the so-called forward and backward probabilities at each step (again see [8] or any other introduction to HMMs). The forward probabilities at a given time can be expressed as the $n$-vector

$$\alpha_t(i) = P(o_1, o_2, \ldots, o_t, q_t = s_i|\lambda),$$

i.e., the probability of the partial observation sequence up until time $t$ and the state at time $t$, given the model. The backward probabilities $\beta_t$ can be expressed analogously and are not considered here. The standard computation is to express the vector $\alpha_t$ recursively as

$$\alpha_{t+1}(j) = b_j(o_{t+1}) \sum_{i=1}^{n} \left( \alpha_t(i) a_{ij} \right).$$

In this form it is readily apparent that computing $\alpha_{t+1}$ from $\alpha_t$ involves $O(n^2)$ operations, as each of the $n$ entries in the vector involves a sum of $n$ terms.

When the transition probabilities are based just on the differences between the underlying coordinates corresponding to the states, $a_{ij} = h(j - i)$, the recursive computation of $\alpha$ becomes

$$\alpha_{t+1}(j) = b_j(o_{t+1}) \sum_{i=1}^{n} \left( \alpha_t(i) h(j - i) \right).$$

The summation term is simply the convolution of $\alpha_t$ with $h$. In general, this discrete convolution can be computed in $O(n \log n)$ time using the FFT. While this is a simple observation, it enables efficient calculation of the forward and backward probabilities for problems where the states are embedded in a grid.

In certain specific cases convolution can be computed in linear time. One case of particular interest is the so-called box sum, in which the convolution kernel is a constant function within a region. That is, $h(j) = k$ over some interval and $h(j) = 0$ outside that interval. A Gaussian can be well approximated by convolution of just a few such box filters [10], and thus it is possible to approximately compute the functions in the first and second rows of Figure 1 in $O(Tn)$ time. Similarly to the Viterbi case, functions can be created from combinations of box-sums. In this case a weighted sum of the individual functions is used rather than their minimum.

## 3  Coin-Tossing Models and Web Usage Analysis

We now turn to the application mentioned in the introduction: using a coin-tossing model with a one-dimensional embedding of states to estimate the download probability of items at a Web site. Our data comes from the Internet Archive site (www.archive.org), which offers digital text, movie, and audio files. Each item on the site has a separate *description* page, which contains the option to download it; this is similar to the paper description pages on CiteSeer or the e-print arXiv and to the item description pages at online retailers (with the option to purchase). On a site of this type, the probability that a user chooses to acquire an item, conditioned on having visited the description page, can be viewed as a measure of interest [1].

This ratio of acquisitions to visits is particularly useful as a way of tracking the *changes* in user interest in an item. Suppose the item is featured prominently on the site; or an active off-site link to the item description drives a new sub-population of users to it; or a technical problem makes it impossible to obtain the item — these are all discrete events that can have a sudden, significant effect on the fraction of users who download the item. By identifying such discrete changes, we can discover the most significant events, both on the site and on the Web at large, that have affected user interest in each item. Such a history of events can be useful to site administrators, as feedback to the users of the site, and for researchers.

This type of change-detection fits naturally into the framework of HMMs. For a fixed item, each observation corresponds to a user's visit to the item description,

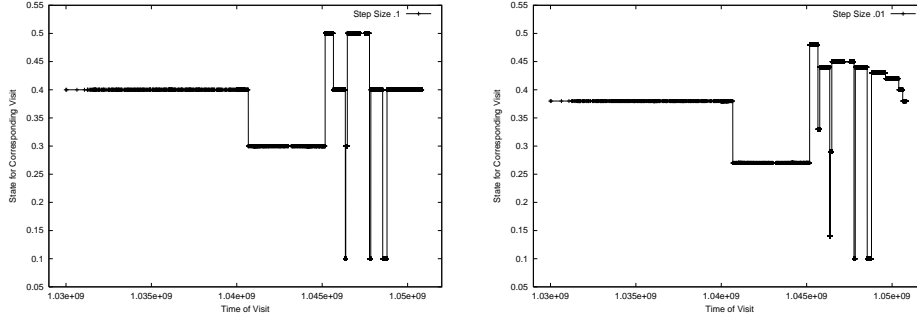

Figure 2: Estimate of underlying download bias; best state sequence for models with step sizes of .1 (9 states) on the left and .01 (81 states) on the right.

and there are two observable symbols $V = \{1, 0\}$, corresponding to the decision to download or not. We assume a model in which there is a hidden coin of some unknown bias that is flipped when the user visits the description and whose outcome determines the download decision. Thus, each state $s_i$ corresponds to a discretized value $p_i$ of the underlying bias parameter. The natural observation cost function $b_i'(k)$ is simply the negative log of the probability $p$ for a head and $(1 - p)$ for a tail.

The points at which state transitions occur in the optimal state sequence thus become candidates for discrete changes in user interest. The form of the state transition costs is based on our assumptions about the nature of these changes. As indicated above, they often result from the introduction of a new sub-population with different interests or expectations; thus, it is natural to expect that the transition cost should rise monotonically as the change in bias increases, but that even large changes should happen with some regularity.

We quantize the underlying bias parameter values equally, such that $|p_i - p_j| \propto |i - j|$ and use a cost function of the form

$$a_{ij}' = \min\left(k_1 |i - j|, k_2 |i - j| + k_3\right),$$

where the $k_i$ are positive constants and $k_1 > k_2$. This two-slope linear model is monotone increasing but once the change in bias becomes large enough the rate of increase is small. The model prefers constant or small changes in bias but allows for arbitrarily large changes, similarly to the "truncated model" common in robust statistics.

Figure 2 shows the best state sequence obtained with the Viterbi algorithm under this model, using two different discretizations of the parameter space, for an input sequence of $11,159$ visits from August 2002 to April 2003 to the description page for a particular video in the Internet Archive. On the left is a 9-state model with probabilities ranging from .1 to .9 in steps of size .1. On the right is an 81-state model with the same range of .1 to .9 but where the steps are of size .01. The $x$-axis shows the visit time (UTC in billions of seconds since the epoch) and the $y$-axis shows the bias associated with the state in the optimal sequence at that time.

We begin by observing that both models capture a number of discrete changes in download behavior. These changes correspond to genuine external events. In particular, both models capture the long-term drop and rebound in bias which corresponds to the time period where the item was highlighted on a top-level page, as well as the two rightmost short downward spikes which correspond to technical problems that made downloads temporarily impossible. Even though these latter

failures were relatively short-lived, lasting a few hours out of the several-month range, they are detected easily by the stochastic model; in contrast, temporal windowing techniques miss such short events.

The two plots, however, exhibit some subtle but important differences that illustrate the qualitatively greater power we obtain from a larger state set. In particular, the 81-state model has four short downward spikes rather than three in the time interval from 1.045 to 1.05. The latter two are the technical failures identified by both models, but the first two correspond to two distinct off-site referring pages each of which drove a significant amount of low-interest user traffic to the item. While the 81-state model was able to resolve these as separate events, the 9-state model blurs them into an artificial period of medium bias, followed by a downward spike to the lowest possible state (i.e. the same state it used for the technical failures). Finally, the 81-state model discovers a gradual decline in the download rate near the end of the plot that is not visible when there are fewer states.

We see that a model with a larger state set is able to pick up the effects of different types of events — both on-site and off-site highlighting of the item, as well as technical problems — and that these events often result in sudden, discrete changes. Moreover, it appears that beyond a certain point, the set of significant events remains roughly fixed even as the resolution in the state set increases. While we do not show the result here, an 801-state model with step size .001 produces a plot that is qualitatively indistinguishable from the 81 state model with step size .01 — only the $y$-values provide more detail with the smaller step size.

# References

[1] J. Aizen, D. Huttenlocher, J. Kleinberg, A. Novak, "Traffic-Based Feedback on the Web," To appear in *Proceedings of the National Academy of Sciences*.

[2] G. Borgefors, "Distance Transformations in Digital Images", *Computer Vision, Graphics and Image Processing*, Vol. 34, pp. 344-371, 1986.

[3] Y. Gil and M. Werman, "Computing 2D Min, Max and Median Filters" *IEEE Trans. PAMI*, Vol. 15, 504-507, 1993.

[4] P. Felzenszwalb, D. Huttenlocher, "Efficient Matching of Pictorial Structures," *Proc. IEEE Computer Vision and Pattern Recognition Conf.*, 2000, pp. 66-73.

[5] A. Karzanov, "Quick algorithm for determining the distances from the points of the given subset of an integer lattice to the points of its complement", *Cybernetics and System Analysis*, 1992. (Translation from the Russian by Julia Komissarchik.)

[6] G. Kitagawa, "Non-Gaussian State Space Modeling of Nonstationary Time Series", *Journal of the American Statistical Association*, 82, pp. 1032-1063, 1987.

[7] J. Pearl, *Probabilistic Reasoning in Intelligent Systems: Networks of Plausible Inference*, Morgan Kaufmann, 1988.

[8] L. Rabiner, "A tutorial on hidden Markov models and selected applications in speech recognition," *Proceedings of the IEEE* Vol. 77(2), pp. 257-286, 1989.

[9] S.L. Scott, P. Smyth, "The Markov Modulated Poisson Process and Markov Poisson Cascade with Applications to Web Traffic Data," *Bayesian Statistics* 7(2003), to appear.

[10] W.M. Wells, "Efficient synthesis of Gaussian filters by cascaded uniform filters", *IEEE Trans. PAMI*, Vol. 8(2), pp. 234-239, 1986.
